# Fast Learning by Bounding Likelihoods in Sigmoid Type Belief Networks

**Tommi Jaakkola**
tommi@psyche.mit.edu

**Lawrence K. Saul**
lksaul@psyche.mit.edu

**Michael I. Jordan**
jordan@psyche.mit.edu

Department of Brain and Cognitive Sciences
Massachusetts Institute of Technology
Cambridge, MA 02139

## Abstract

Sigmoid type belief networks, a class of probabilistic neural networks, provide a natural framework for compactly representing probabilistic information in a variety of unsupervised and supervised learning problems. Often the parameters used in these networks need to be learned from examples. Unfortunately, estimating the parameters via exact probabilistic calculations (i.e, the EM-algorithm) is intractable even for networks with fairly small numbers of hidden units. We propose to avoid the infeasibility of the E step by bounding likelihoods instead of computing them exactly. We introduce extended and complementary representations for these networks and show that the estimation of the network parameters can be made fast (reduced to quadratic optimization) by performing the estimation in either of the alternative domains. The complementary networks can be used for continuous density estimation as well.

## 1 Introduction

The appeal of probabilistic networks for knowledge representation, inference, and learning (Pearl, 1988) derives both from the sound Bayesian framework and from the explicit representation of dependencies among the network variables which allows ready incorporation of prior information into the design of the network. The Bayesian formalism permits full propagation of probabilistic information across the network regardless of which variables in the network are instantiated. In this sense these networks can be "inverted" probabilistically.

This inversion, however, relies heavily on the use of look-up table representations

of conditional probabilities or representations equivalent to them for modeling dependencies between the variables. For sparse dependency structures such as trees or chains this poses no difficulty. In more realistic cases of reasonably interdependent variables the exact algorithms developed for these belief networks (Lauritzen & Spiegelhalter, 1988) become infeasible due to the exponential growth in the size of the conditional probability tables needed to store the exact dependencies. Therefore the use of compact representations to model probabilistic interactions is unavoidable in large problems. As belief network models move away from tables, however, the representations can be harder to assess from expert knowledge and the important role of learning is further emphasized.

Compact representations of interactions between simple units have long been emphasized in neural networks. Lacking a thorough probabilistic interpretation, however, classical feed-forward neural networks cannot be inverted in the above sense; e.g. given the output pattern of a feed-forward neural network it is not feasible to compute a probability distribution over the possible input patterns that would have resulted in the observed output. On the other hand, stochastic neural networks such as Boltzman machines admit probabilistic interpretations and therefore, at least in principle, can be inverted and used as a basis for inference and learning in the presence of uncertainty.

Sigmoid belief networks (Neal, 1992) form a subclass of probabilistic neural networks where the activation function has a sigmoidal form – usually the logistic function. Neal (1992) proposed a learning algorithm for these networks which can be viewed as an improvement of the algorithm for Boltzmann machines. Recently Hinton et al. (1995) introduced the wake-sleep algorithm for layered bi-directional probabilistic networks. This algorithm relies on forward sampling and has an appealing coding theoretic motivation. The Helmholtz machine (Dayan et al., 1995), on the other hand, can be seen as an alternative technique for these architectures that avoids Gibbs sampling altogether. Dayan et al. also introduced the important idea of bounding likelihoods instead of computing them exactly. Saul et al. (1995) subsequently derived rigorous mean field bounds for the likelihoods. In this paper we introduce the idea of alternative – extended and complementary – representations of these networks by reinterpreting the nonlinearities in the activation function. We show that deriving likelihood bounds in the new representational domains leads to efficient (quadratic) estimation procedures for the network parameters.

## 2   The probability representations

Belief networks represent the joint probability of a set of variables $\{S\}$ as a product of conditional probabilities given by

$$P(S_1, \ldots, S_n) = \prod_{k=1}^{n} P(S_k | pa[k]), \qquad (1)$$

where the notation $pa[k]$, "parents of $S_k$", refers to all the variables that directly influence the probability of $S_k$ taking on a particular value (for equivalent representations, see Lauritzen et al. 1988). The fact that the joint probability can be written in the above form implies that there are no "cycles" in the network; i.e. there exists an ordering of the variables in the network such that no variable directly influences any preceding variables.

In this paper we consider sigmoid belief networks where the variables $S$ are binary

$(0/1)$, the conditional probabilities have the form

$$P(S_i|\text{pa}[i]) = g\Big((2S_i - 1)\sum_j W_{ij}S_j\Big) \tag{2}$$

and the weights $W_{ij}$ are zero unless $S_j$ is a parent of $S_i$, thus preserving the feed-forward directionality of the network. For notational convenience we have assumed the existence of a bias variable whose value is clamped to one. The activation function $g(\cdot)$ is chosen to be the cumulative Gaussian distribution function given by

$$g(x) = \frac{1}{\sqrt{2\pi}}\int_{-\infty}^{x} e^{-\frac{1}{2}z^2}dz = \frac{1}{\sqrt{2\pi}}\int_{0}^{\infty} e^{-\frac{1}{2}(z-x)^2}dz \tag{3}$$

Although very similar to the standard logistic function, this activation function derives a number of advantages from its integral representation. In particular, we may reinterpret the integration as a marginalization and thereby obtain alternative representations for the network. We consider two such representations.

We derive an *extended* representation by making explicit the nonlinearities in the activation function. More precisely,

$$
\begin{aligned}
P(S_i|\text{pa}[i]) &= g\Big((2S_i - 1)\sum_j W_{ij}S_j\Big) \\
&= \int_0^\infty \frac{1}{\sqrt{2\pi}} e^{-\frac{1}{2}[Z_i - (2S_i-1)\sum_j W_{ij}S_j]^2} dZ_i \\
&\stackrel{def}{=} \int_0^\infty P(S_i, Z_i|\text{pa}[i])dZ_i
\end{aligned} \tag{4}
$$

This suggests defining the extended network in terms of the new conditional probabilities $P(S_i, Z_i|\text{pa}[i])$. By construction then the original binary network is obtained by marginalizing over the extra variables $Z$. In this sense the extended network is (marginally) equivalent to the binary network.

We distinguish a *complementary* representation from the extended one by writing the probabilities entirely in terms of continuous variables[1]. Such a representation can be obtained from the extended network by a simple transformation of variables. The new continuous variables are defined by $\tilde{Z}_i = (2S_i - 1)Z_i$, or, equivalently, by $Z_i = |\tilde{Z}_i|$ and $S_i = \theta(\tilde{Z}_i)$ where $\theta(\cdot)$ is the step function. Performing this transformation yields

$$P(\tilde{Z}_i|\text{pa}[i]) = \frac{1}{\sqrt{2\pi}} e^{-\frac{1}{2}[\tilde{Z}_i - \sum_j W_{ij}\theta(\tilde{Z}_j)]^2} \tag{5}$$

which defines a network of conditionally Gaussian variables. The original network in this case can be recovered by conditional marginalization over $\tilde{Z}$ where the conditioning variables are $\theta(\tilde{Z})$.

Figure 1 below summarizes the relationships between the different representations. As will become clear later, working with the alternative representations instead of the original binary representation can lead to more flexible and efficient (least-squares) parameter estimation.

## 3   The learning problem

We consider the problem of learning the parameters of the network from instantiations of variables contained in a training set. Such instantiations, however, need not

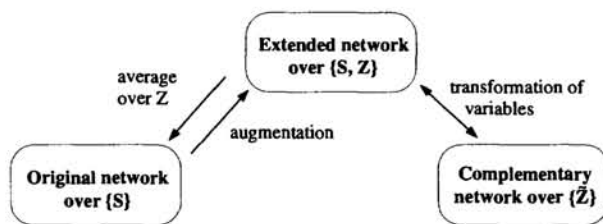

Figure 1: The relationship between the alternative representations.

be complete; there may be variables that have no value assignments in the training set as well as variables that are always instantiated. The tacit division between hidden (H) and visible (V) variables therefore depends on the particular training example considered and is not an intrinsic property of the network.

To learn from these instantiations we adopt the principle of maximum likelihood to estimate the weights in the network. In essence, this is a density estimation problem where the weights are chosen so as to match the probabilistic behavior of the network with the observed activities in the training set. Central to this estimation is the ability to compute likelihoods (or log-likelihoods) for any (partial) configuration of variables appearing in the training set. In other words, if we let $X^V$ be the configuration of visible or instantiated variables[2] and $X^H$ denote the hidden or uninstantiated variables, we need to compute marginal probabilities of the form

$$\log P(X^V) = \log \sum_{X^H} P(X^V, X^H) \tag{6}$$

If the training samples are independent, then these log marginals can be added to give the overall log-likelihood of the training set

$$\log P(\text{training set}) = \sum_t \log P(X^{V_t}) \tag{7}$$

Unfortunately, computing each of these marginal probabilities involves summing (integrating) over an exponential number of different configurations assumed by the hidden variables in the network. This renders the sum (integration) intractable in all but few special cases (e.g. trees and chains). It is possible, however, to instead find a manageable lower bound on the log-likelihood and optimize the weights in the network so as to maximize this bound.

To obtain such a lower bound we resort to Jensen's inequality:

$$
\begin{aligned}
\log P(X^V) &= \log \sum_{X^H} P(X^H, X^V) = \log \sum_{X^H} Q(X^H) \frac{P(X^H, X^V)}{Q(X^H)} \\
&\geq \sum_{X^H} Q(X^H) \log \frac{P(X^H, X^V)}{Q(X^H)}
\end{aligned} \tag{8}
$$

Although this bound holds for all distributions $Q(X)$ over the hidden variables, the accuracy of the bound is determined by how closely $Q$ approximates the posterior distribution $P(X^H|X^V)$ in terms of the Kullback-Leibler divergence; if the approximation is perfect the divergence is zero and the inequality is satisfied with equality. Suitable choices for $Q$ can make the bound both accurate and easy to compute. The feasibility of finding such $Q$, however, is highly dependent on the choice of the representation for the network.

## 4  Likelihood bounds in different representations

To complete the derivation of the likelihood bound (equation 8) we need to fix the representation for the network. Which representation to select, however, affects the quality and accuracy of the bound. In addition, the accompanying bound of the chosen representation implies bounds in the other two representational domains as they all code the same distributions over the observables. In this section we illustrate these points by deriving bounds in the complementary and extended representations and discuss the corresponding bounds in the original binary domain.

Now, to obtain a lower bound we need to specify the approximate posterior $Q$. In the complementary representation the conditional probabilities are Gaussians and therefore a reasonable approximation (mean field) is found by choosing the posterior approximation from the family of factorized Gaussians:

$$Q(\tilde{Z}) = \prod_i \frac{1}{\sqrt{2\pi}} e^{-(\tilde{Z}_i - h_i)^2/2} \tag{9}$$

Substituting this into equation 8 we obtain the bound

$$\log P(S^*) \geq -\frac{1}{2}\sum_i (h_i - \Sigma_j J_{ij} g(h_j))^2 - \frac{1}{2}\sum_{ij} J_{ij}^2 g(h_j)g(-h_j) \tag{10}$$

The means $h_i$ for the hidden variables are adjustable parameters that can be tuned to make the bound as tight as possible. For the instantiated variables we need to enforce the constraints $g(h_i) = S_i^*$ to respect the instantiation. These can be satisfied very accurately by setting $h_i = 4(2S_i^* - 1)$. A very convenient property of this bound and the complementary representation in general is the quadratic weight dependence – a property very conducive to fast learning. Finally, we note that the complementary representation transforms the binary estimation problem into a continuous density estimation problem.

We now turn to the interpretation of the above bound in the binary domain. The same bound can be obtained by first fixing the inputs to all the units to be the means $h_i$ and then computing the negative total mean squared error between the fixed inputs and the corresponding probabilistic inputs propagated from the parents. The fact that this procedure in fact gives a lower bound on the log-likelihood would be more difficult to justify by working with the binary representation alone.

In the extended representation the probability distribution for $Z_i$ is a truncated Gaussian given $S_i$ and its parents. We therefore propose the partially factorized posterior approximation:

$$Q(S, Z) = \prod_i Q(Z_i|S_i)Q(S_i) \tag{11}$$

where $Q(Z_i|S_i)$ is a truncated Gaussian:

$$Q(Z_i|S_i) = \frac{1}{g((2S_i-1)h_i)} \frac{1}{\sqrt{2\pi}} e^{-\frac{1}{2}(Z_i - (2S_i-1)h_i)^2} \tag{12}$$

As in the complementary domain the resulting bound depends quadratically on the weights. Instead of writing out the bound here, however, it is more informative to see its derivation in the binary domain.

A factorized posterior approximation (mean field) $Q(S) = \prod_i q_i^{S_i}(1-q_i)^{1-S_i}$ for the binary network yields a bound

$$\log P(S^*) \geq \sum_i \left\{ \langle S_i \log g(\textstyle\sum_j J_{ij}s_j)\rangle + \langle(1-S_i)\log(1-g(\textstyle\sum_j J_{ij}s_j))\rangle \right\}$$

$$-\sum_i [q_i \log q_i + (1 - q_i) \log(1 - q_i)] \tag{13}$$

where the averages $\langle \cdot \rangle$ are with respect to the $Q$ distribution. These averages, however, do not conform to analytical expressions. The tractable posterior approximation in the extended domain avoids the problem by implicitly making the following Legendre transformation:

$$\log g(x) = [\frac{1}{2}x^2 + \log g(x)] - \frac{1}{2}x^2 \geq \lambda x - G(\lambda) - \frac{1}{2}x^2 \tag{14}$$

which holds since $x^2/2 + \log g(x)$ is a convex function. Inserting this back into the relevant parts of equation 13 and performing the averages gives

$$\begin{aligned}
\log P(S^*) \geq & \sum_i \Big\{ [q_i \lambda_i - (1 - q_i)\bar{\lambda}_i] \sum_j J_{ij} q_j - q_i G(\lambda_i) - (1 - q_i)G(\bar{\lambda}_i) \Big\} \\
& - \frac{1}{2}(\sum_j J_{ij} q_j)^2 - \frac{1}{2}\sum_{ij} J_{ij}^2 q_j (1 - g_j) \\
& - \sum_i [q_i \log q_i + (1 - q_i) \log(1 - q_i)]
\end{aligned} \tag{15}$$

which is quadratic in the weights as expected. The mean activities $q$ for the hidden variables and the parameters $\lambda$ can be optimized to make the bound tight. For the instantiated variables we set $q_i = S_i^*$.

## 5 Numerical experiments

To test these techniques in practice we applied the complementary network to the problem of detecting motor failures from spectra obtained during motor operation (see Petsche et al. 1995). We cast the problem as a continuous density estimation problem. The training set consisted of 800 out of 1283 FFT spectra each with 319 components measured from an electric motor in a good operating condition but under varying loads. The test set included the remaining 483 FFTs from the same motor in a good condition in addition to three sets of 1340 FFTs each measured when a particular fault was present. The goal was to use the likelihood of a test FFT with respect to the estimated density to determine whether there was a fault present in the motor.

We used a layered $6 \rightarrow 20 \rightarrow 319$ generative model to estimate the training set density. The resulting classification error rates on the test set are shown in figure 2 as a function of the threshold likelihood. The achieved error rates are comparable to those of Petsche et al. (1995).

## 6 Conclusions

Network models that admit probabilistic formulations derive a number of advantages from probability theory. Moving away from explicit representations of dependencies, however, can make these properties harder to exploit in practice. We showed that an efficient estimation procedure can be derived for sigmoid belief networks, where standard methods are intractable in all but a few special cases (e.g. trees and chains). The efficiency of our approach derived from the combination of two ideas. First, we avoided the intractability of computing likelihoods in these networks by computing lower bounds instead. Second, we introduced new representations for these networks and showed how the lower bounds in the new representational domains transform the parameter estimation problem into

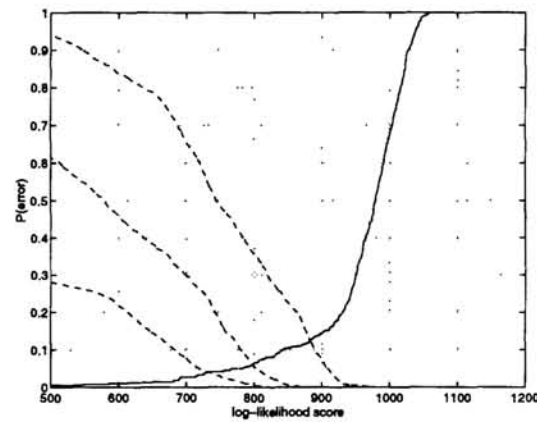

Figure 2: The probability of error curves for missing a fault (dashed lines) and misclassifying a good motor (solid line) as a function of the likelihood threshold.

quadratic optimization.

## Acknowledgments

The authors wish to thank Peter Dayan for helpful comments. This project was supported in part by NSF grant CDA-9404932, by a grant from the McDonnell-Pew Foundation, by a grant from ATR Human Information Processing Research Laboratories, by a grant from Siemens Corporation, and by grant N00014-94-1-0777 from the Office of Naval Research. Michael I. Jordan is a NSF Presidential Young Investigator.

## Footnotes

[1]While the binary variables are the outputs of each unit the continuous variables pertain to the inputs – hence the name complementary.

[2]To postpone the issue of representation we use $X$ to denote $S$, $\{S, Z\}$, or $\tilde{Z}$ depending on the particular representation chosen.

## References

P. Dayan, G. Hinton, R. Neal, and R. Zemel (1995). The helmholtz machine. *Neural Computation* **7**: 889-904.

A. Dempster, N. Laird, and D. Rubin. Maximum likelihood from incomplete data via the EM algorithm (1977). *J. Roy. Statist. Soc. B* **39**:1-38.

G. Hinton, P. Dayan, B. Frey, and R. Neal (1995). The wake-sleep algorithm for unsupervised neural networks. *Science* **268**: 1158-1161.

S. L. Lauritzen and D. J. Spiegelhalter (1988). Local computations with probabilities on graphical structures and their application to expert systems. *J. Roy. Statist. Soc. B* **50**:154-227.

R. Neal. Connectionist learning of belief networks (1992). *Artificial Intelligence* **56**: 71-113.

J. Pearl (1988). *Probabilistic Reasoning in Intelligent Systems.* Morgan Kaufmann: San Mateo.

T. Petsche, A. Marcantonio, C. Darken, S. J. Hanson, G. M. Kuhn, I. Santoso (1995). A neural network autoassociator for induction motor failure prediction. In *Advances in Neural Information Processing Systems 8.* MIT Press.

L. K. Saul, T. Jaakkola, and M. I. Jordan (1995). Mean field theory for sigmoid belief networks. *M.I.T. Computational Cognitive Science Technical Report* **9501**.